# Invariant Object Recognition Using a Distributed Associative Memory

Harry Wechsler and George Lee Zimmerman
Department of Electrical Engineering
University of Minnesota
Minneapolis, MN 55455

## Abstract

This paper describes an approach to 2-dimensional object recognition. Complex-log conformal mapping is combined with a distributed associative memory to create a system which recognizes objects regardless of changes in rotation or scale. Recalled information from the memorized database is used to classify an object, reconstruct the memorized version of the object, and estimate the magnitude of changes in scale or rotation. The system response is resistant to moderate amounts of noise and occlusion. Several experiments, using real, gray scale images, are presented to show the feasibility of our approach.

## Introduction

The challenge of the visual recognition problem stems from the fact that the projection of an object onto an image can be confounded by several dimensions of variability such as uncertain perspective, changing orientation and scale, sensor noise, occlusion, and non-uniform illumination. A vision system must not only be able to sense the identity of an object despite this variability, but must also be able to characterize such variability -- because the variability inherently carries much of the valuable information about the world. Our goal is to derive the functional characteristics of image representations suitable for invariant recognition using a distributed associative memory. The main question is that of finding appropriate transformations such that interactions between the internal structure of the resulting representations and the distributed associative memory yield invariant recognition. As Simon [1] points out, all mathematical derivation can be viewed simply as a change of representation, making evident what was previously true but obscure. This view can be extended to all problem solving. Solving a problem then means transforming it so as to make the solution transparent.

We approach the problem of object recognition with three requirements: classification, reconstruction, and characterization. Classification implies the ability to distinguish objects that were previously encountered. Reconstruction is the process by which memorized images can be drawn from memory given a distorted version exists at the input. Characterization involves extracting information about how the object has changed from the way in which it was memorized. Our goal in this paper is to discuss a system which is able to recognize memorized 2-dimensional objects regardless of geometric distortions like changes in scale and orientation, and can characterize those transformations. The system also allows for noise and occlusion and is tolerant of memory faults.

The following sections, Invariant Representation and Distributed Associative Memory, respectively, describe the various components of the system in detail. The Experiments section presents the results from several experiments we have performed on real data. The paper concludes with a discussion of our results and their implications for future research.

## 1. Invariant Representation

The goal of this section is to examine the various components used to produce the vectors which are associated in the distributed associative memory. The block diagram which describes the various functional units involved in obtaining an invariant image representation is shown in Figure 1. The image is complex-log conformally mapped so that rotation and scale changes become translation in the transform domain. Along with the conformal mapping, the image is also filtered by a space variant filter to reduce the effects of aliasing. The conformally mapped image is then processed through a Laplacian in order to solve some problems associated with the conformal mapping. The Fourier transform of both the conformally mapped image and the Laplacian processed image produce the four output vectors. The magnitude output vector $|\bullet|_1$ is invariant to linear transformations of the object in the input image. The phase output vector $\Phi_2$ contains information concerning the spatial properties of the object in the input image.

### 1.1 Complex-Log Mapping and Space Variant Filtering

The first box of the block diagram given in Figure 1 consists of two components: Complex-log mapping and space variant filtering. Complex-log mapping transforms an image from rectangular coordinates to polar exponential coordinates. This transformation changes rotation and scale into translation. If the image is mapped onto a complex plane then each pixel (x,y) on the Cartesian plane can be described mathematically by $z = x + jy$. The complex-log mapped points w are described by

$$w = \ln(z) = \ln(|z|) + j\theta_z \qquad (1)$$

where $|z| = (x^2 + y^2)^{\frac{1}{2}}$ and $\theta_z = \tan^{-1}(y/x)$.

Our system sampled 256x256 pixel images to construct 64x64 complex-log mapped images. Samples were taken along radial lines spaced 5.6 degrees apart. Along each radial line the step size between samples increased by powers of 1.08. These numbers are derived from the number of pixels in the original image and the number of samples in the complex-log mapped image. An excellent examination of the different conditions involved in selecting the appropriate number of samples for a complex-log mapped image is given in [2]. The non-linear sampling can be split into two distinct parts along each radial line. Toward the center of the image the samples are dense enough that no anti-aliasing filter is needed. Samples taken at the edge of the image are large and an anti-aliasing filter is necessary. The image filtered in this manner has a circular region around the center which corresponds to an area of highest resolution. The size of this region is a function of the number of angular samples and radial samples. The filtering is done, at the same time as the sampling, by convolving truncated Bessel functions with the image in the space domain. The width of the Bessel functions main lobe is inversely proportional to the eccentricity of the sample point.

A problem associated with the complex-log mapping is sensitivity to center misalignment of the sampled image. Small shifts from the center causes dramatic distortions in the complex-log mapped image. Our system assumes that the object is centered in the image frame. Slight misalignments are considered noise. Large misalignments are considered as translations and could be accounted for by changing the gaze in such a way as to bring the object into the center of the frame. The decision about what to bring into the center of the frame is an active function and should be determined by the task. An example of a system which could be used to guide the translation process was developed by Anderson and Burt [3]. Their pyramid system analyzes the input image at different tem-

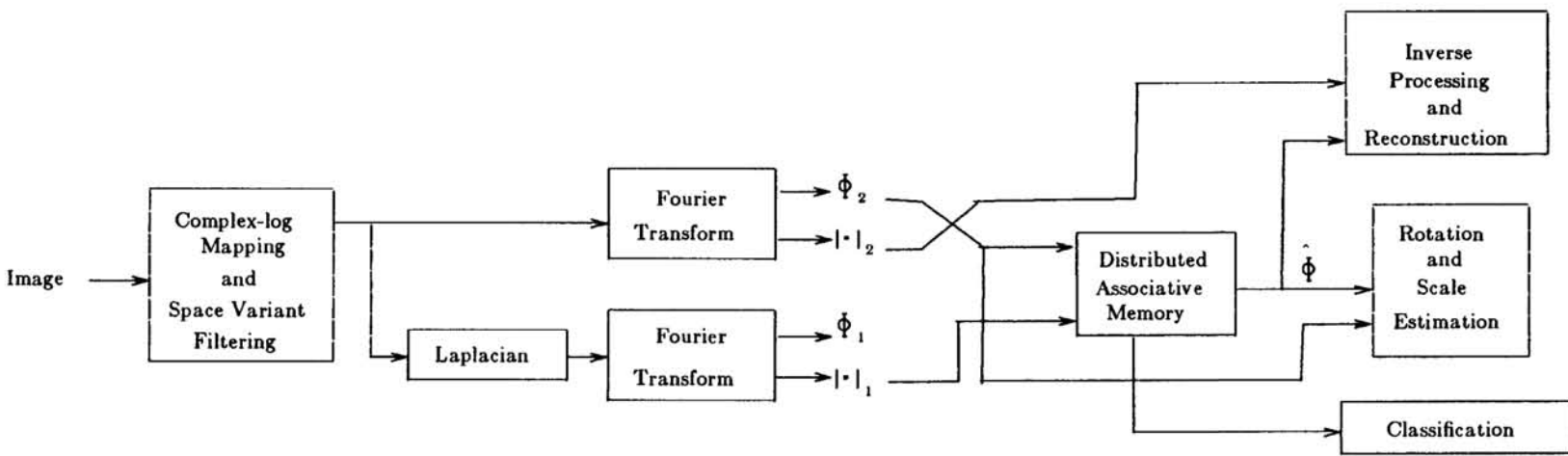

Figure 1. Block Diagram of the System.

poral and spatial resolution levels. Their smart sensor was then able to shift its fixation such that interesting parts of the image (ie. something large and moving) was brought into the central part of the frame for recognition.

## 1.2 Fourier Transform

The second box in the block diagram of Figure 1 is the Fourier transform. The Fourier transform of a 2-dimensional image f(x,y) is given by

$$F(u,v) = \int_{-\infty}^{\infty} \int_{-\infty}^{\infty} f(x,y) e^{-j(ux+vy)} \, dx \, dy \qquad (2)$$

and can be described by two 2-dimensional functions corresponding to the magnitude $|F(u,v)|$ and phase $\Phi_F(u,v)$. The magnitude component of the Fourier transform which is invariant to translation, carries much of the contrast information of the image. The phase component of the Fourier transform carries information about how things are placed in an image. Translation of f(x,y) corresponds to the addition of a linear phase component. The complex-log mapping transforms rotation and scale into translation and the magnitude of the Fourier transform is invariant to those translations so that $|\bullet|_1$ will not change significantly with rotation and scale of the object in the image.

## 1.3 Laplacian

The Laplacian that we use is a difference-of-Gaussians (DOG) approximation to the $\nabla^2 G$ function as given by Marr [4].

$$\nabla^2 G = \frac{1}{\pi\sigma^4} [1 - r^2/2\sigma^2] \, e^{\{-r^2/2\sigma^2\}} \qquad (3)$$

The result of convolving the Laplacian with an image can be viewed as a two step process. The image is blurred by a Gaussian kernel of a specified width $\sigma$. Then the isotropic second derivative of the blurred image is computed. The width of the Gaussian kernel is chosen such that the conformally mapped image is visible -- approximately 2 pixels in our experiments. The Laplacian sharpens the edges of the object in the image and sets any region that did not change much to zero. Below we describe the benefits from using the Laplacian.

The Laplacian eliminates the stretching problem encountered by the complex-log mapping due to changes in object size. When an object is expanded the complex-log mapped image will translate. The pixels vacated by this translation will be filled with more pixels sampled from the center of the scaled object. These new pixels will not be significantly different than the displaced pixels so the result looks like a stretching in the complex-log mapped image. The Laplacian of the complex-log mapped image will set the new pixels to zero because they do not significantly change from their surrounding pixels. The Laplacian eliminates high frequency spreading due to the finite structure of the discrete Fourier transform and enhances the differences between memorized objects by accentuating edges and de-emphasizing areas of little change.

## 2. Distributed Associative Memory (DAM)

The particular form of distributed associative memory that we deal with in this paper is a memory matrix which modifies the flow of information. Stimulus vectors are associated with response vectors and the result of this association is spread over the entire memory space. Distributing in this manner means that information about a small portion of the association can be found in a large area of the memory. New associations are placed

over the older ones and are allowed to interact. This means that the size of the memory matrix stays the same regardless of the number of associations that have been memorized. Because the associations are allowed to interact with each other an implicit representation of structural relationships and contextual information can develop, and as a consequence a very rich level of interactions can be captured. There are few restrictions on what vectors can be associated there can exist extensive indexing and cross-referencing in the memory. Distributed associative memory captures a distributed representation which is context dependent. This is quite different from the simplistic behavioral model [5].

The *construction* stage assumes that there are n pairs of m-dimensional vectors that are to be associated by the distributed associative memory. This can be written as

$$M\vec{s_i} = \vec{r_i} \quad \text{for } i = 1,...,n \tag{4}$$

where $\vec{s_i}$ denotes the $i^{th}$ stimulus vector and $\vec{r_i}$ denotes the $i^{th}$ corresponding response vector. We want to construct a memory matrix $M$ such that when the $k^{th}$ stimulus vector $\vec{s_k}$ is projected onto the space defined by M the resulting projection will be the corresponding response vector $\vec{r_k}$. More specifically we want to solve the following equation:

$$MS = R \tag{5}$$

where $S = [\ \vec{s_1} \mid \vec{s_2} \mid ... \mid \vec{s_n}\ ]$ and $R = [\ \vec{r_1} \mid \vec{r_2} \mid ... \mid \vec{r_n}\ ]$. A unique solution for this equation does not necessarily exist for any arbitrary group of associations that might be chosen. Usually, the number of associations $n$ is smaller than $m$, the length of the vector to be associated, so the system of equations is underconstrained. The constraint used to solve for a unique matrix M is that of minimizing the square error, $\|MS - R\|^2$, which results in the solution

$$M = RS^+ \tag{6}$$

where $S^+$ is known as the Moore-Penrose generalized inverse of S [6].

The *recall* operation projects an unknown stimulus vector $\tilde{s}$ onto the memory space M. The resulting projection yields the response vector $\tilde{r}$

$$\tilde{r} = M\tilde{s} \tag{7}$$

If the memorized stimulus vectors are independent and the unknown stimulus vector $\tilde{s}$ is one of the memorized vectors $\vec{s_k}$, then the recalled vector will be the associated response vector $\vec{r_k}$. If the memorized stimulus vectors are dependent, then the vector recalled by one of the memorized stimulus vectors will contain the associated response vector and some *crosstalk* from the other stored response vectors.

The recall can be viewed as the weighted sum of the response vectors. The recall begins by assigning weights according to how well the unknown stimulus vector matches with the memorized stimulus vector using a linear least squares classifier. The response vectors are multiplied by the weights and summed together to build the recalled response vector. The recalled response vector is usually dominated by the memorized response vector that is closest to the unknown stimulus vector.

Assume that there are n associations in the memory and each of the associated stimulus and response vectors have m elements. This means that the memory matrix has $m^2$ elements. Also assume that the noise that is added to each element of a memorized

stimulus vector is independent, zero mean, with a variance of $\sigma_i^2$. The recall from the memory is then

$$\vec{r} = \vec{r_k} + \vec{v}_o = M(\vec{s_k} + \vec{v_i}) = \vec{r_k} + M\vec{v_i} \qquad (8)$$

where $\vec{v_i}$ is the input noise vector and $\vec{v}_o$ is the output noise vector. The ratio of the average output noise variance to the average input noise variance is

$$\sigma_o^2/\sigma_i^2 = \frac{1}{m}\mathrm{Tr}[MM^T] \qquad (9)$$

For the autoassociative case this simplifies to

$$\sigma_o^2/\sigma_i^2 = \frac{n}{m} \qquad (10)$$

This says that when a noisy version of a memorized input vector is applied to the memory the recall is improved by a factor corresponding to the ratio of the number of memorized vectors to the number of elements in the vectors. For the heteroassociative memory matrix a similar formula holds as long as n is less than m [7].

$$\sigma_o^2/\sigma_i^2 = \frac{1}{m}\mathrm{Tr}[RR^T]\mathrm{Tr}[(S^TS)^{-1}] \qquad (11)$$

Fault tolerance is a byproduct of the distributed nature and error correcting capabilities of the distributed associative memory. By distributing the information, no single memory cell carries a significant portion of the information critical to the overall performance of the memory.

## 3. Experiments

In this section we discuss the result of computer simulations of our system. Images of objects are first preprocessed through the subsystem outlined in section 2. The output of such a subsystem is four vectors: $|\bullet|_1$, $\Phi_1$, $|\bullet|_2$, and $\Phi_2$. We construct the memory by associating the stimulus vector $|\bullet|_1$ with the response vector $\Phi_2$ for each object in the database. To perform a recall from the memory the unknown image is preprocessed by the same subsystem to produce the vectors $|\bullet|_1$, $\Phi_1$, $|\bullet|_2$, and $\Phi_2$. The resulting stimulus vector $|\bullet|_1$ is projected onto the memory matrix to produce a response vector which is an estimate of the memorized phase $\Phi_2$. The estimated phase vector $\Phi_2$ and the magnitude $|\bullet|_1$ are used to reconstruct the memorized object. The difference between the estimated phase $\Phi_2$ and the unknown phase $\Phi_2$ is used to estimate the amount of rotation and scale experienced by the object.

The database of images consists of twelve objects: four keys, four mechanical parts, and four leaves. The objects were chosen for their essentially two-dimensional structure. Each object was photographed using a digitizing video camera against a black background. We emphasize that all of the images used in creating and testing the recognition system were taken at different times using various camera rotations and distances. The images are digitized to 256x256, eight bit quantized pixels, and each object covers an area of about 40x40 pixels. This small object size relative to the background is necessary due to the non-linear sampling of the complex-log mapping. The objects were centered within the frame by hand. This is the source of much of the noise and could have been done automatically using the object's center of mass or some other criteria determined by the task. The orientation of each memorized object was arbitrarily chosen such that their major axis

was vertical. The 2-dimensional images that are the output from the invariant representation subsystem are scanned horizontally to form the vectors for memorization. The database used for these experiments is shown in Figure 2.

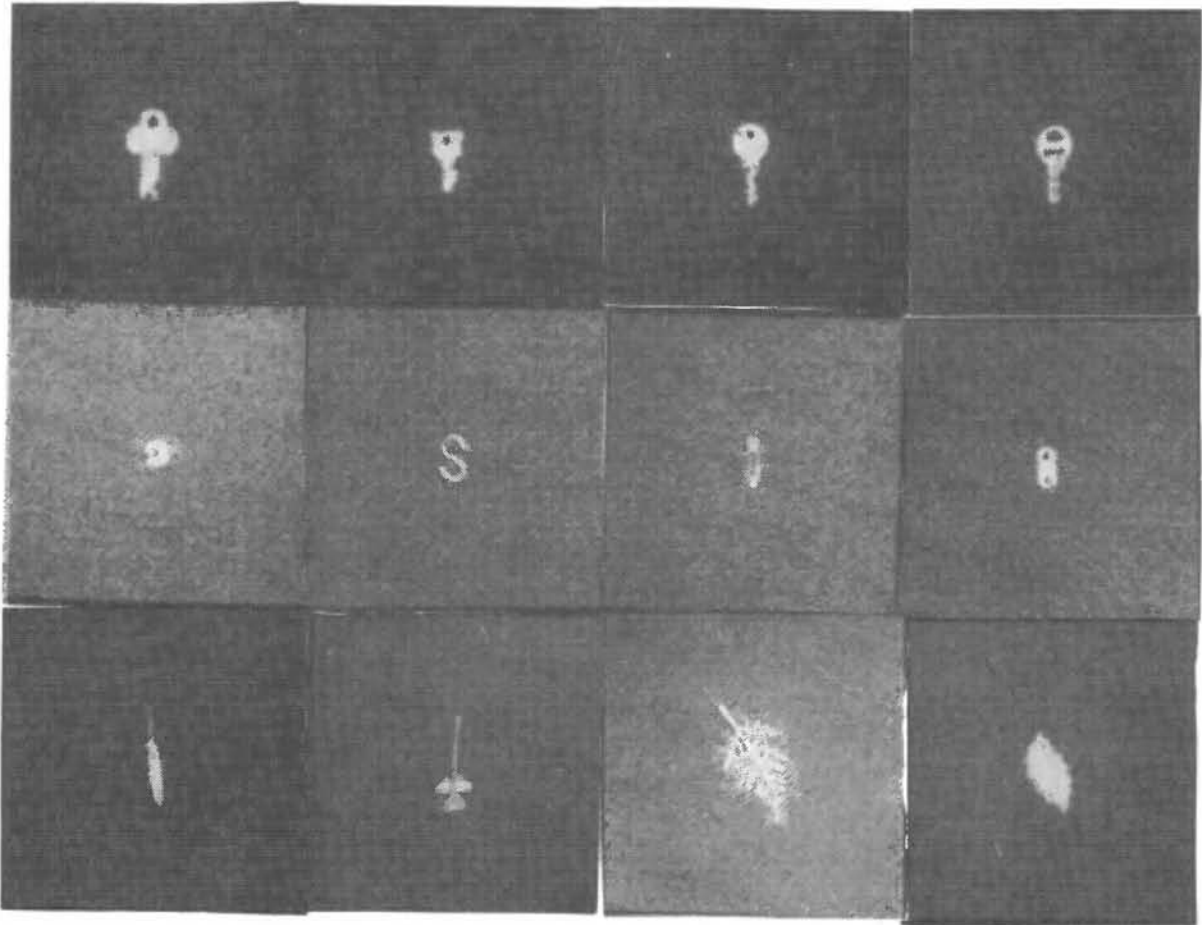

Figure 2. The Database of Objects Used in the Experiments

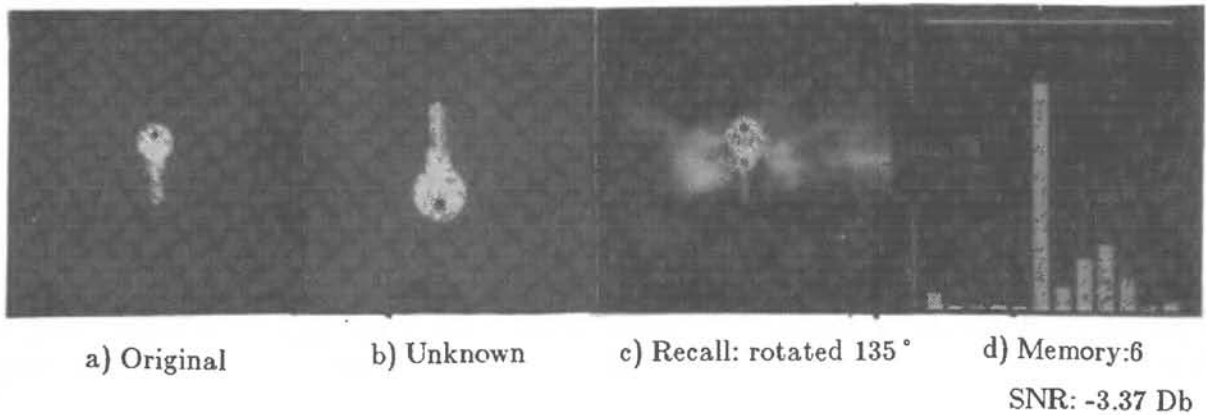

a) Original      b) Unknown      c) Recall: rotated 135°      d) Memory:6

SNR: -3.37 Db

Figure 3. Recall Using a Rotated and scaled key

The first example of the operation of our system is shown in Figure 3. Figure 3a) is the image of one of the keys as it was memorized. Figure 3b) is the unknown object presented to our system. The unknown object in this case is the same key that has been rotated by 180 degrees and scaled. Figure 3c) is the recalled, reconstructed image. The

rounded edges of the recalled image are artifacts of the complex-log mapping. Notice that the reconstructed recall is the unrotated memorized key with some noise caused by errors in the recalled phase. Figure 3d) is a histogram which graphically displays the classification vector which corresponds to $S^+s$. The histogram shows the interplay between the memorized images and the unknown image. The "6" on the bargraph indicates which of the twelve classes the unknown object belongs. The histogram gives a value which is the best linear estimate of the image relative to the memorized objects. Another measure, the signal-to-noise ratio (SNR), is given at the bottom of the recalled image. SNR compares the variance of the ideal recall after processing with the variance of the difference between the ideal and actual recall. This is a measure of the amount of noise in the recall. The SNR does not carry much information about the quality of the recall image because the noise measured by the SNR is due to many factors such as misalignment of the center, changing reflections, and dependence between other memorized objects -- each affecting quality in a variety of ways. Rotation and scale estimates are made using a vector $D$ corresponding to the difference between the unknown vector $\Phi_2$ and the recalled vector $\Phi_2$. In an ideal situation $D$ will be a plane whose gradient indicates the exact amount of rotation and scale the recalled object has experienced. In our system the recalled vector $\Phi_2$ is corrupted with noise which means rotation and scale have to be estimated. The estimate is made by letting the first order difference $D$ at each point in the plane vote for a specified range of rotation or scale.

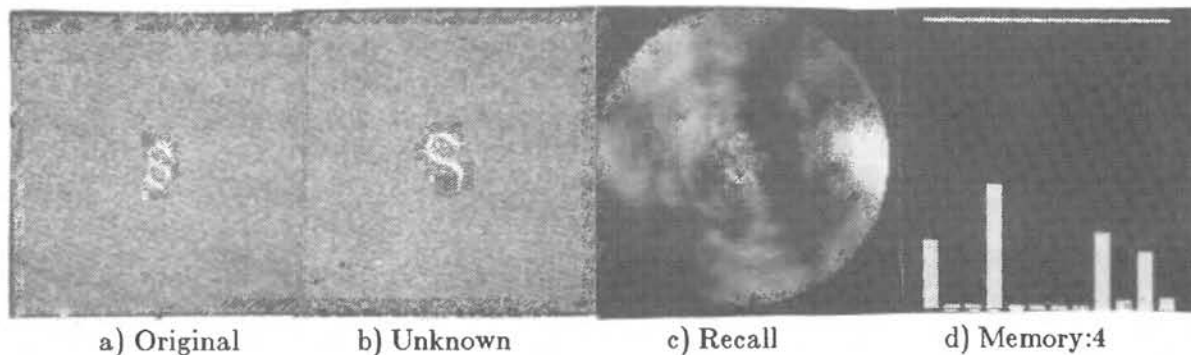

a) Original      b) Unknown      c) Recall      d) Memory:4

Figure 4 Recall Using Scaled and Rotated "S" with Occlusion

Figure 4 is an example of occlusion. The unknown object in this case is an "S" curve which is larger and slightly tilted from the memorized "S" curve. A portion of the bottom curve was occluded. The resulting reconstruction is very noisy but has filled in the missing part of the bottom curve. The noisy recall is reflected in both the SNR and the interplay between the memories shown by the histogram.

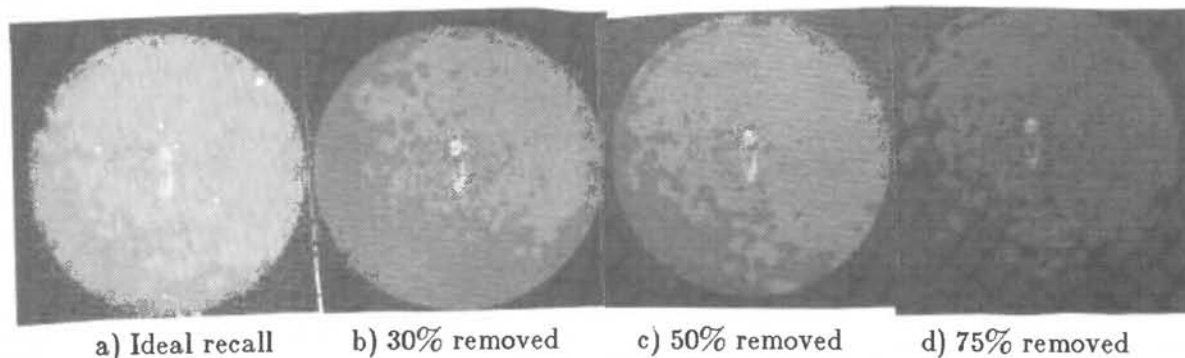

a) Ideal recall      b) 30% removed      c) 50% removed      d) 75% removed

Figure 5. Recall for Memory Matrix Randomly Set to Zero

Figure 5 is the result of *randomly* setting the elements of the memory matrix to

zero. Figure 5a) shows is the ideal recall. Figure 5b) is the recall after 30 percent of the memory matrix has been set to zero. Figure 5c) is the recall for 50 percent and Figure 5d) is the recall for 75 percent. Even when 90 percent of the memory matrix has been set to zero a faint outline of the pin could still be seen in the recall. This result is important in two ways. First, it shows that the distributed associative memory is robust in the presence of noise. Second, it shows that a completely connected network is not necessary and as a consequence a scheme for data compression of the memory matrix could be found.

## 4. Conclusion

In this paper we demonstrate a computer vision system which recognizes 2-dimensional objects invariant to rotation or scale. The system combines an invariant representation of the input images with a distributed associative memory such that objects can be classified, reconstructed, and characterized. The distributed associative memory is resistant to moderate amounts of noise and occlusion. Several experiments, demonstrating the ability of our computer vision system to operate on real, grey scale images, were presented.

Neural network models, of which the distributed associative memory is one example, were originally developed to simulate biological memory. They are characterized by a large number of highly interconnected simple processors which operate in parallel. An excellent review of the many neural network models is given in [8]. The distributed associative memory we use is linear, and as a result there are certain desirable properties which will not be exhibited by our computer vision system. For example, feedback through our system will not improve recall from the memory. Recall could be improved if a non-linear element, such as a sigmoid function, is introduced into the feedback loop. Non-linear neural networks, such as those proposed by Hopfield [9] or Anderson et. al. [10], can achieve this type of improvement because each memorized pattern is associated with stable points in an energy space. The price to be paid for the introduction of non-linearities into a memory system is that the system will be difficult to analyze and can be unstable. Implementing our computer vision system using non-linear distributed associative memory is a goal of our future research.

We are presently extending our work toward 3-dimensional object recognition. Much of the present research in 3-dimensional object recognition is limited to polyhedral, non-occluded objects· in a clean, highly controlled environment. Most systems are edge based and use a generate-and-test paradigm to estimate the position and orientation of recognized objects. We propose to use an approach based on characteristic views [11] or aspects [12] which suggests that the infinite 2-dimensional projections of a 3-dimensional object can be grouped into a finite number of topological equivalence classes. An efficient 3-dimensional recognition system would require a parallel indexing method to search for object models in the presence of geometric distortions, noise, and occlusion. Our object recognition system using distributed associative memory can fulfill those requirements with respect to characteristic views.

## References

[1] Simon, H. A., (1984), **The Science of the Artificial (2nd ed.),** MIT Press.
[2] Massone, L., G. Sandini, and V. Tagliasco (1985), "Form-invariant" topological mapping strategy for 2D shape recognition, **CVGIP,** 30, 169-188.
[3] Anderson, C. H., P. J. Burt, and G. S. Van Der Wal (1985), Change detection and tracking using pyramid transform techniques, **Proc. of the SPIE Conference on Intelligence, Robots, and Computer Vision,** Vol. 579, 72-78.

[4] Marr, D. (1982), **Vision,** W. H. Freeman, 1982.

[5] Hebb, O. D. (1949), **The Organization of Behavior,** New York: Wiley.

[6] Kohonen, T. (1984), **Self-Organization and Associative-Memories,** Springer-Verlag.

[7] Stiles, G. S. and D. L. Denq (1985), On the effect of noise on the Moore-Penrose generalized inverse associative memory, **IEEE Trans. on PAMI,** 7, 3, 358-360.

[8] M^cClelland, J. L., and D. E. Rumelhart, and the PDP Research Group (Eds.) (1986), **Parallel Distributed, Processing,** Vol. 1, 2, MIT Press.

[9] Hopfield, J. J. (1982), Neural networks and physical systems with emergent collective computational abilities, **Proc. Natl. Acad. Sci. USA,** 79, April 1982.

[10] Anderson, J. A., J. W. Silverstein, S. A. Ritz, and R. S. Jones (1977), Distinctive features, categorical perception, and probability learning: some applications of a neural model, **Psychol. Rev.,** 84,413-451.

[11] Chakravarty, I., and H. Freeman (1982), Characteristic views as a basis for 3-D object recognition, **Proc. SPIE on Robot Vision,** 336, 37-45.

[12] Koenderink, J. J., and A. J. Van Doorn (1979), Internal representation of solid shape with respect to vision, **Biol. Cybern.,** 32,4,211-216.
